# A Machine Learning Approach to Conjoint Analysis

**Olivier Chapelle, Zaïd Harchaoui**
Max Planck Institute for Biological Cybernetics
Spemannstr. 38 - 72076 Tübingen - Germany
{olivier.chapelle,zaid.harchaoui}@tuebingen.mpg.de

## Abstract

Choice-based conjoint analysis builds models of consumer preferences over products with answers gathered in questionnaires. Our main goal is to bring tools from the machine learning community to solve this problem more efficiently. Thus, we propose two algorithms to quickly and accurately estimate consumer preferences.

## 1 Introduction

Conjoint analysis (also called trade-off analysis) is one of the most popular marketing research technique used to determine which features a new product should have, by *conjointly* measuring consumers trade-offs between discretized[1] attributes. In this paper, we will focus on the choice-based conjoint analysis (CBC) framework [11] since it is both widely used and realistic: at each question in the survey, the consumer is asked to choose one product from several.

The preferences of a consumer are modeled via a *utility* function representing how much a consumer likes a given product. The utility $u(\mathbf{x})$ of a product $\mathbf{x}$ is assumed to be the sum of the partial utilities (or *partworths*) for each attribute, i.e. linear: $u(\mathbf{x}) = \mathbf{w} \cdot \mathbf{x}$. However, instead of observing pairs $(\mathbf{x}_l, y_l)$, the training samples are of the form $(\{\mathbf{x}_k^1, \ldots, \mathbf{x}_k^p\}, y_k)$ indicating that among the $p$ products $\{\mathbf{x}_k^1, \ldots, \mathbf{x}_k^p\}$, the $y_k^{th}$ was preferred. Without noise, this is expressed mathematically by $u(\mathbf{x}_k^{y_k}) \geq u(\mathbf{x}_k^b), \quad \forall\, b \neq y_k$.

Let us settle down the general framework of a regular conjoint analysis survey. We have a population of $n$ consumers available for the survey. The survey consists of a questionnaire of $q$ questions for each consumer, each asking to choose one product from a basket of $p$. Each product profile is described through $a$ attributes with $l_1, ..., l_a$ levels each, *via* a vector of length $m = \sum_{s=1}^{a} l_s$, with 1 at positions of levels taken by each attribute and 0 elsewhere.

Marketing researchers are interested in estimating individual partworths in order to perform for instance a segmentation of the population afterwards. But traditional conjoint estimation techniques are not reliable for this task since the number of parameters $m$ to be estimated is usually larger than the number of answers $q$ available for each consumer. They estimate instead the partworths on the whole population (aggregated partworths). Here we

aim to investigate this issue, for which machine learning can provide efficient tools. We also address adaptive questionnaire design with active learning heuristics.

## 2 Hierarchical Bayes Analysis

The main idea of HB[2] is to estimate the individual utility functions under the constraint that their variance should not be too small. By doing so, the estimation problem is not ill-posed and the lack of information for a consumer can be completed by the other ones.

### 2.1 Probabilistic model

In this section, we follow [11] for the description of the HB model and its implementation. This method aims at estimating the individual *linear* utility functions $u_i(\mathbf{x}) = \mathbf{w}_i \cdot \mathbf{x}$, for $1 \le i \le n$. The probabilistic model is the following:

1. The individual partworths $\mathbf{w}_i$ are drawn from a Gaussian distribution with mean $\boldsymbol{\alpha}$ (representing the aggregated partworths) and covariance $\Sigma$ (encoding population's heterogeneity),

2. The covariance matrix $\Sigma$ has an invert Wishart prior, and $\boldsymbol{\alpha}$ has an (improper) flat prior.

3. Given a set of products $(\mathbf{x}_1, \ldots \mathbf{x}_p)$, the probability that the consumer $i$ chooses the product $\mathbf{x}^*$ is given by

$$P(\mathbf{x}^* | \mathbf{w}_i) = \frac{\exp(\mathbf{w}_i \cdot \mathbf{x}^*)}{\sum_{b=1}^{p} \exp(\mathbf{w}_i \cdot \mathbf{x}_b)}. \tag{1}$$

### 2.2 Model estimation

We describe now the standard way of estimating $\boldsymbol{\alpha}$, $\mathbf{w} \equiv (\mathbf{w}_1, \ldots, \mathbf{w}_n)$ and $\Sigma$ based on Gibbs sampling and then propose a much faster algorithm that approximates the *maximum a posteriori* (MAP) solution.

**Gibbs sampling**    As far as we know, all implementations of HB rely on a variant of the Gibbs sampling [11]. During one iteration, each of the three sets of variables ($\boldsymbol{\alpha}$, $\mathbf{w}$ and $\Sigma$) is drawn in turn from its posterior distribution the two others being fixed. Sampling for $\boldsymbol{\alpha}$ and $\Sigma$ is straightforward, whereas sampling from $P(\mathbf{w}|\boldsymbol{\alpha}, \Sigma, Y) \propto P(Y|\mathbf{w})$. $P(\mathbf{w}|\boldsymbol{\alpha}, \Sigma)$ is achieved with the Metropolis-Hastings algorithm.

When convergence is reached, the sampling goes on and finally outputs the empirical expectation of $\boldsymbol{\alpha}$, $\mathbf{w}$ and $\Sigma$. Although the results of this sampling-based implementation of HB[3] are impressive, practitioners complain about its computational burden.

**Approximate MAP solution**    So far HB implementations make predictions by evaluating (1) at the empirical mean of the samples, in contrast with the standard bayesian approach, which would average the rhs of (1) over the different samples, given samples $\mathbf{w}$ from the posterior. In order to alleviate the computational issues associated with Gibbs sampling, we suggest to consider the maximum of the posterior distribution (*maximum a posteriori*, MAP) rather than its mean.

To find $\boldsymbol{\alpha}$, $\mathbf{w}$ and $\Sigma$ which maximize $P(\boldsymbol{\alpha}, \mathbf{w}, \Sigma | Y)$, let us use Bayes' rule,

$$\begin{aligned} P(\boldsymbol{\alpha}, \mathbf{w}, \Sigma | Y) &\propto P(Y | \boldsymbol{\alpha}, \mathbf{w}, \Sigma) \cdot P(\mathbf{w} | \boldsymbol{\alpha}, \Sigma) \cdot P(\boldsymbol{\alpha} | \Sigma) \cdot P(\Sigma) \\ &\propto P(Y | \mathbf{w}) \cdot P(\mathbf{w} | \boldsymbol{\alpha}, \Sigma) \cdot P(\Sigma) \end{aligned} \tag{2}$$

Maximizing (2) with respect to $\Sigma$ yields $\Sigma_{\text{MAP}} = \frac{I + C_{\mathbf{w}}}{n+d}$, with $C_{\mathbf{w}}$ being the "covariance" matrix of the $\mathbf{w}_i$ centered at $\boldsymbol{\alpha}$: $C_{\mathbf{w}} = \sum (\mathbf{w}_i - \boldsymbol{\alpha})(\mathbf{w}_i - \boldsymbol{\alpha})^\top$. Putting back this value in (2), we get

$$-\log P(\boldsymbol{\alpha}, \mathbf{w}, \Sigma_{\text{MAP}} | Y) = -\log P(Y | \mathbf{w}) + \log |I + C_{\mathbf{w}}(\boldsymbol{\alpha})| + C, \tag{3}$$

where $C$ is an irrelevant constant. Using the model (1), the first term in the rhs of (3) is convex in $\mathbf{w}$, but not the second term. For this reason, we propose to change $\log |I + C_{\mathbf{w}}|$ by $\text{trace}(C_{\mathbf{w}}) = \sum ||\mathbf{w}_i - \boldsymbol{\alpha}||^2$ (this would be a valid approximation if $\text{trace}(C_{\mathbf{w}}) \ll 1$). With this new prior on $\mathbf{w}$, the rhs of (3) becomes

$$W(\boldsymbol{\alpha}, \mathbf{w}) = \sum_{i=1}^{n} -\log P(Y_i | \mathbf{w}_i) + ||\mathbf{w}_i - \boldsymbol{\alpha}||^2. \tag{4}$$

As in equation (3), this objective function is minimized with respect to $\boldsymbol{\alpha}$ when $\boldsymbol{\alpha}$ is equal to the empirical mean of the $\mathbf{w}_i$. We thus suggest the following iterative scheme to minimize the convex functional (4):

1. For a given $\boldsymbol{\alpha}$, minimize (4) with respect to each of the $\mathbf{w}_i$ independently.
2. For a given $\mathbf{w}$, set $\boldsymbol{\alpha}$ to the empirical mean[4] of the $\mathbf{w}$.

Thanks to the convexity, this optimization problem can be solved very efficiently. A Newton approach in step 1, as well as in step 2 to speed-up the global convergence to a fixed point $\boldsymbol{\alpha}$, has been implemented. Only couple of steps in both cases are necessary to reach convergence.

**Remark** The approximation from equation (3) to (4) might be too crude. After all it boils down to setting $\Sigma$ to the identity matrix. One might instead consider $\Sigma$ as an hyperparameter and optimize it by maximizing the marginalized likelihood [14].

## 3 Conjoint Analysis with Support Vector Machines

Similarly to what has recently been proposed in [3], we are now investigating the use of Support Vector Machines (SVM) [1, 12] to solve the conjoint estimation problem.

### 3.1 Soft margin formulation of conjoint estimation

Let us recall the learning problem. At the $k$-th question, the consumer chooses the $y_k^{th}$ product from the basket $\{\mathbf{x}_k^1, \ldots, \mathbf{x}_k^p\}$: $\mathbf{w} \cdot \mathbf{x}_k^{y_k} \geq \mathbf{w} \cdot \mathbf{x}_k^b$, $\forall b \neq y_k$. Our goal is to estimate the individual partworths $\mathbf{w}$, with the individual utility function now being $u(\mathbf{x}) = \mathbf{w} \cdot \mathbf{x}$. With a reordering of the products, we can actually suppose that $y_k = 1$. Then the above inequalities can be rewritten as a set of $p - 1$ constraints:

$$\mathbf{w} \cdot (\mathbf{x}_k^1 - \mathbf{x}_k^b) \geq 0, \quad 2 \leq b \leq p. \tag{5}$$

Eq. (5) shows that the conjoint estimation problem can be cast as a classification problem in the product-profiles differences space. From this point of view, it seems quite natural to use state-of-the-art classifiers such as SVMs for this purpose.

More specifically, we propose to train a $L_2$-soft margin classifier (see also [3] for a similar approach) with only positive examples and with a hyperplane passing through the origin (no bias), modelling the noise in the answers with slack variables $\xi_{kb}$:

$$\left\{ \begin{array}{ll} \text{Minimize} & \mathbf{w}^2 + C \sum_{k=1}^{q} \sum_{b=2}^{p} \xi_{kb}^2 \\ \text{subject to} & \mathbf{w} \cdot (\mathbf{x}_k^1 - \mathbf{x}_k^b) \geq 1 - \xi_{kb}. \end{array} \right.$$

### 3.2 Estimation of individual utilities

It was proposed in [3] to train one SVM per consumer to get $\mathbf{w}_i$ and to compute the individual partworths by regularizing with the aggregated partworths $\overline{\mathbf{w}} = \frac{1}{n} \sum_{i=1}^{n} \mathbf{w}_i$: $\mathbf{w}_i^* = \frac{\mathbf{w}_i + \overline{\mathbf{w}}}{2}$.

Instead, to estimate the individual utility partworths $\mathbf{w}_i$, we suggest the following optimization problem (the set $Q_i$ contains the indices $j$ such that the consumer $i$ was asked to choose between products $\mathbf{x}_k^1, \ldots, \mathbf{x}_k^p$) :

$$\left\{ \begin{array}{l} \text{Minimize} \;\; \mathbf{w}_i^2 + \frac{C}{q_i} \sum_{k \in Q_i} \sum_{b=2}^{p} \xi_{kb}^2 + \frac{\tilde{C}}{\sum_{j \neq i} q_j} \sum_{k \notin Q_i} \sum_{b=2}^{p} \xi_{kb}^2 \\ \text{subject to} \;\; \mathbf{w}_i \cdot (\mathbf{x}_k^1 - \mathbf{x}_k^b) \geq 1 - \xi_{kb}, \quad \forall k, \; \forall b \geq 2 \,. \end{array} \right.$$

Here the ratio $\frac{C}{\tilde{C}}$ determines the trade-off between the individual scale and the aggregated one.[5] For $\frac{C}{\tilde{C}} = 1$, the population is modeled as if it were homogeneous, i.e. all partworths $\mathbf{w}_i$ are equal. For $\frac{C}{\tilde{C}} \gg 1$, the individual partworths are computed independently, without taking into account aggregated partworths.

## 4 Related work

**Ordinal regression** Very recently [2] explores the so-called *ordinal regression* task for ranking, and derive two techniques for hyperparameters learning and model selection in a hierarchical bayesian framework, Laplace approximation and Expectation Propagation respectively. Ordinal regression is similar yet distinct from conjoint estimation since training data are supposed to be rankings or ratings in contrast with conjoint estimation where training data are choice-based. See [4] for more extensive bibliography.

**Large margin classifiers** Casting the preference problem in a classification framework, leading to learning by convex optimization, was known for a long time in the psychometrics community. [5] pioneered the use of large margin classifiers for ranking tasks. [3] introduced the kernel methods machinery for conjoint analysis on the individual scale. Very recently [10] proposes an alternate method for dealing with heterogeneity in conjoint analysis, which boils down to a very similar optimization to our HB-MAP approximation objective function, but with large margin regularization and with minimum deviation from the aggregated partworths.

**Collaborative filtering** Collaborative filtering exploits similarity between ratings across a population. The goal is to predict a person's rating on new products given the person's past ratings on similar products and the ratings of other people on all the products. Again collaborative is designed for overlapping training samples for each consumer, and usually rating/ranking training data, whereas conjoint estimation usually deals with different questionnaires for each consumer and choice-based training data.

# 5  Experiments

**Artificial experiments**    We tested our algorithms on the same benchmarking artificial experimental setup used in [3, 16]. The simulated product profiles consist of 4 attributes, each of them being discretized through 4 levels. A random design was used for the questionnaire. For each question, the consumer was asked to choose one product from a basket of 4. A population of 100 consumers was simulated, each of them having to answer 4 questions. Finally, the results presented below are averaged over 5 trials.

The 100 true consumer partworths were generated from a Gaussian distribution with mean $(-\beta, -\beta/3, \beta/3, \beta)$ (for each attribute) and with a diagonal covariance matrix $\sigma^2 I$. Each answer is a choice from the basket of products, sampled from the discrete logit-type distribution (1). Hence when $\beta$ (called the *magnitude*[6]) is large, the consumer will choose with high probability the product with the highest utility, whereas when $\beta$ is small, the answers will be less reliable. The ratio $\sigma^2/\beta$ controls the *heterogeneity*[7] of the population.

Finally, as in [3], the performances are computed using the mean of the $L_2$ distances between the true and estimated individual partworths (also called RMSE). Beforehand the partworths are translated such that the mean on each attribute is 0 and normalized to 1.

**Real experiments**    We tested our algorithms on disguised industrial datasets kindly provided by *Sawtooth Software Inc.*, the world leading company in conjoint analysis softwares.

11 one-choice-based[8] conjoint surveys datasets[9] were used for real experiments below. The number of attributes ranged from 3 to 6 (hence total number of levels from 13 to 28), the size of the baskets, to pick one product from at each question, ranged from 2 to 5, and the number of questions ranged from 6 to 15. The numbers of respondents ranged roughly from 50 to 1200. Since here we did not address the issue of no choice options in question answering, we removed[10] questions where customers refused to choose a product from the basket and chose the no-choice-option as an answer[11].

Finally, as in [16], the performances are computed using the *hit rate*, i.e. the misprediction rate of the preferred product.

## 5.1   Analysis of HB-MAP

We compare in this section our implementation of the HB method described in Section 2, that we call HB-MAP, to HB-S, the standard HB implementation.

The average training time for HB-S was 19 minutes (with 12000 iterations as suggested in [11]), whereas our implementation based on the approximation of the MAP solution took in average only 1.8 seconds. So our primary goal, i.e. to alleviate the sampling phase complexity, was achieved since we got *a speed-up factor of the order of 1000*.

The accuracy does not seem to be significantly weakened by this new implementation. Indeed, as shown in both Table 1 and Table 2, the performances achieved by HB-MAP were surprisingly often as good as HB-S's, and sometimes even a bit better. This might be

explained by the fact that assuming that the covariance matrix is quasi-diagonal is a reasonable approximation, and that the mode of the posterior distribution is actually roughly close to the mean, for the real datasets considered. Additionally it is likely that HB-S may have demanded much more iterations for convergence to systematically behave more accurately than HB-MAP as one would have normally expected.

## 5.2 Analysis of SVMs

We now turn to the SVM approach presented in section 3.2 that we call Im.SV[12]. We did not use a non-linear kernel in our experiments. Hence it was possible to minimize (3.2) directly in the primal, instead of using the dual formulation as done usually. This turned out to be faster since the number of constraints was, for our problem, larger than the number of variables. The resulting mean training time was 4.7 seconds. The so-called *chapspan*, span estimate of leave-one-out prediction error [17], was used to select a suitable value of $C$[13], since it gave a quasi-convex estimation on the regularization path.

The performances of Im.SV in Table 2, compared to the HB methods and logistic regression [3] are very satisfactory in case of artificial experiments. In real experiments, Im.SV gives overall quite satisfactory results, but sometimes disappointing ones in Table 2. One reason might be that hyperparameters $(C, \tilde{C})$ were optimized once for the whole population. This may also be due to the lack of robustness[14] of Im.SV to heterogeneity in the number of training samples for each consumer.

Table 1: Average RMSE between estimated and true individual partworths

| Mag | Het | HB-S | HB-MAP | Logistic | Im.SV |
|---|---|---|---|---|---|
| L | L | 0.90 | 0.83 | 0.84 | 0.86 |
| L | H | 0.95 | 0.91 | 1.16 | 0.90 |
| H | L | 0.44 | 0.40 | 0.43 | 0.41 |
| H | H | 0.72 | 0.68 | 0.82 | 0.67 |

Table 2: Hit rate performances on real datasets.

| | Im.SV | HB-MAP | HB-S | | Im.SV | HB-MAP | HB-S |
|---|---|---|---|---|---|---|---|
| $Dat1_2$ | 0.16 | 0.16 | 0.17 | $Dat1_5$ | 0.52 | 0.45 | 0.48 |
| $Dat2_2$ | 0.15 | 0.13 | 0.15 | $Dat2_5$ | 0.58 | 0.47 | 0.51 |
| | Im.SV | HB-MAP | HB-S | | Im.SV | HB-MAP | HB-S |
| $Dat1_3$ | 0.37 | 0.24 | 0.25 | | | | |
| $Dat2_3$ | 0.34 | 0.33 | 0.33 | $Dat1_4$ | 0.33 | 0.36 | 0.35 |
| $Dat3_3$ | 0.35 | 0.28 | 0.24 | $Dat2_4$ | 0.33 | 0.36 | 0.28 |
| $Dat4_3$ | 0.35 | 0.31 | 0.28 | $Dat3_4$ | 0.45 | 0.40 | 0.25 |

**Legend of Tables 1 and 2** The first two columns indicate the **Mag**nitude and the **Het**erogeneity (**H**igh or **L**ow). $p$ in $Datm_p$ is the number of products respondents are asked to choose one from at each question.

# 6 Active learning

**Motivation**  Traditional experimental designs are built by minimizing the variance of an estimator (e.g. orthogonal designs [6]). However, they are sub-optimal because they do not take into account the previous answers of the consumer. Therefore *adaptive conjoint analysis* was proposed [11, 16] for adaptively designing questionnaires.

The adaptive design concept is often called *active learning* in machine learning, as the algorithm can actively select questions whose responses are likely to be informative. In the SVM context, a common and intuitive strategy is to select, as the next point to be labeled, the nearest one from the decision boundary (see for instance [15]).

**Experiments**  We implemented this heuristic for conjoint analysis by selecting for each question a set of products whose estimated utilities are as close as possible[15]. To compare the different designs, we used the same artificial simulations as in section 5, but with 16 questions per consumer in order to fairly compare to the orthogonal design.

Table 3: Comparison of the RMSE achieved by different designs.

| Mag | Het | Random | Orthogonal | Adaptive |
|-----|-----|--------|------------|----------|
| L | L | 0.66 | **0.61** | 0.66 |
| L | H | 0.62 | 0.56 | 0.56 |
| H | L | 0.31 | 0.29 | **0.24** |
| H | H | 0.49 | 0.45 | **0.34** |

Results in Table 3 show that active learning produced an adaptive design which seems efficient, especially in the case of high magnitude, i.e. when the answers are not noisy[16].

# 7 Discussion

We may need to capture correlations between attributes to model interaction effects among them. The polynomial kernel $K(u, v) = (u.v + 1)^d$ seems particularly relevant for such a task. HB methods kernelization can be done in the framework presented in [7]. For large margin methods [10, 3] give a way to use the *kernel trick* in the space of product-profiles differences. Prior knowledge of product-profile structure [3] may also be incorporated in the estimation process by using *virtual examples* [12].

[9] approach would allow us to improve our approximate MAP solution by learning a variational approximation of a non-isotropic diagonal covariance matrix.
A fully bayesian HB setting, i.e. with a maximum likelihood type II[17] (ML II) step, in contrast of sampling from the posterior, is known in the statistics community as bayesian multinomial logistic regression. [18] use Laplace approximation to compute integration over hyperparameters for multi-class classification, while [8] develop a variational approximation of the posterior distribution.
New insights on learning gaussian process regression in a HB framework have just been given in [13], where a method combining an EM algorithm and a generalized Nyström approximation of covariance matrix is proposed, and could be incorporated in the HB-MAP approximation above.

## 8   Conclusion

Choice-based conjoint analysis seems to be a very promising application field for machine learning techniques. Further research include fully bayesian HB methods, extensions to non-linear models as well as more elaborate and realistic active learning schemes.

## Acknowledgments

The authors are very grateful to J. Quiñonero-Candela and C. Rasmussen for fruitful discussions, and O. Toubia for providing us with his HB implementation. Many thanks to *Sawtooth Software Inc.* for providing us with real conjoint analysis datasets.

## Footnotes

[1] e.g. if the discretized attribute is *weight*, the levels would be light/heavy.

[2]Technical papers of *Sawtooth software* [11], the world leading company for conjoint analysis softwares, provide very useful and extensive references.

[3]that we will call HB-Sampled or HB-S in the rest of the paper.

[4] which is consistent with the $L_2$-loss measuring deviations of $\mathbf{w}_i$-s from $\boldsymbol{\alpha}$.

[5]$C \geq \tilde{C}$ In this way, directions for which the $\mathbf{x}_j$, $j \in Q_i$ contain information are estimated accurately, whereas the others directions are estimated thanks to the answers of the other consumers.

[6]as in [3], we tested **H**igh **Mag**nitude ($\beta = 3$) and **L**ow **Mag**nitude ($\beta = 0.5$).

[7]It was either set to $\sigma^2 = 3\beta$ or $\sigma^2 = 0.5\beta$, respectively **H**igh and **L**ow **Het**erogeneity cases.

[8]We limited ourselves to datasets in which respondents were asked to choose 1 product among a basket at each question.

[9]see [4] for more details on the numerical features of the datasets.

[10]One could use EM-based methods to deal with such missing training choice data.

[11]When this procedure boiled down to unreasonable number of questions for hold-out evaluation of our algorithms, we simply removed the corresponding individuals.

[12]since individual choice data are **Im**mersed in the rest of the population choice data, *via* the optimization objective

[13]We observed that the value of the constant $\tilde{C}$ was irrelevant, and that only the ratio $C/\tilde{C}$ mattered.

[14]Indeed the no-choice data cleaning step might have lead to a strong unbalance to which Im.SV is maybe much more sensitive than HB-MAP or HB-S.

[15]Since the bottom-line goal of the conjoint analysis is not really to estimate the partworths but to design the "optimal" product, adaptive design can also be helpful by focusing on products which have a high estimated utility.

[16]Indeed noisy answers are neither informative nor reliable for selecting the next question.

[17]aka evidence maximization or hyperparameters learning

## References

[1] B. E. Boser, I. M. Guyon, and V. N. Vapnik. A training algorithm for optimal margin classifiers. In *Proc. 5th Annu. Workshop on Comput. Learning Theory*, 1992.

[2] W. Chu and Z. Ghahramani. Gaussian processes for ordinal regression. Technical report, University College London, 2004.

[3] T. Evgeniou, C. Boussios, and G. Zacharia. Generalized robust conjoint estimation. *Marketing Science*, 25, 2005.

[4] Z. Harchaoui. Statistical learning approaches to conjoint estimation. Technical report, Max Planck Institute for Biological Cybernetics, to appear.

[5] R. Herbrich, T. Graepel, and K. Obermayer. Large margin rank boundaries for ordinal regression. In *Advances in Large Margin Classifiers*. MIT Press, 2000.

[6] J. Huber and K. Zwerina. The importance of utility balance in efficient choice designs. *Journal of Marketing Research*, 33, 1996.

[7] T. S. Jaakkola and D. Haussler. Probabilistic kernel regression models. In *Artificial Intelligence and Statistics*, 1999.

[8] T. S. Jaakkola and M. I. Jordan. Bayesian logistic regression: a variational approach. *Statistics and Computing*, 10:25–37, 2000.

[9] T. Jebara. Convex invariance learning. In *Artificial Intelligence and Statistics*, 2003.

[10] C. A. Micchelli and M. Pontil. Kernels for multi–task learning. In *Advances in Neural Information Processing Systems 17*, 2005.

[11] Sawtooth Software. *Research paper series*. Available at www.sawtoothsoftware.com/techpap.shtml#hbrel.

[12] B. Schölkopf and A. Smola. *Learning with kernels*. MIT Press, 2002.

[13] A. Schwaighofer, V. Tresp, and K. Yu. Hierarchical bayesian modelling with gaussian processes. In *Advances in Neural Information Processing Systems 17*, 2005.

[14] M. Tipping. Bayesian inference: Principles and practice. In *Advanced Lectures on Machine Learning*. Springer, 2004.

[15] S. Tong and D. Koller. Support vector machine active learning with applications to text classification. In *Journal of Machine Learning Research*, volume 2, 2001.

[16] O. Toubia, J. R. Hauser, and D. I. Simester. Polyhedral methods for adaptive choice-based conjoint analysis. *Journal of Marketing Research*, 41(1):116–131, 2004.

[17] V. Vapnik and O. Chapelle. Bounds on error expectation for support vector machines. *Neural Computation*, 12(9), 2000.

[18] C. K. I. Williams and D. Barber. Bayesian classification with gaussian processes. *IEEE Trans. Pattern Anal. Mach. Intell.*, 20, 1998.
